# Convergence of Optimistic and Incremental Q-Learning

Eyal Even-Dar*                    Yishay Mansour†

## Abstract

We show the convergence of two deterministic variants of Q-learning. The first is the widely used optimistic Q-learning, which initializes the Q-values to large initial values and then follows a greedy policy with respect to the Q-values. We show that setting the initial value sufficiently large guarantees the converges to an $\epsilon$-optimal policy. The second is a new and novel algorithm *incremental Q-learning*, which gradually promotes the values of actions that are not taken. We show that incremental Q-learning converges, in the limit, to the optimal policy. Our incremental Q-learning algorithm can be viewed as derandomization of the $\epsilon$-greedy Q-learning.

## 1  Introduction

One of the challenges of Reinforcement Learning is learning in an unknown environment. The environment is modeled by an MDP and we can only observe the trajectory of states, actions and rewards generated by the agent wandering in the MDP. There are two basic conceptual approaches to the learning problem. The first is model base, where we first reconstruct a model of the MDP, and then find an optimal policy for the approximate model. Recently polynomial time algorithms have been developed for this approach, initially in [7] and latter extended in [3]. The second are direct methods that update their estimated policy after each step. The most popular of the direct methods is Q-learning [13].

Q-learning uses the information observed to approximate the optimal value function, from which one can construct an optimal policy. There are various proofs that Q-learning converges, in the limit, to the optimal value function, under very mild conditions [1, 11, 12, 8, 6, 2]. In a recent result the convergence rates of Q-learning are computed and an interesting dependence on the learning rates is exhibited [4].

Q-learning is an off-policy that can be run on top of any strategy. Although, it is an off policy algorithm, in many cases its estimated value function is used to guide the selection of actions. Being always greedy with respect to the value function may result in poor performance, due to the lack of exploration, and often randomization is used guarantee proper exploration.

We show the convergence of two deterministic strategies. The first is *optimistic*

---

*School of Computer Science, Tel-Aviv University, Tel-Aviv, Israel. *evend@cs.tau.ac.il*
†School of Computer Science, Tel-Aviv University, Israel. *mansour@cs.tau.ac.il*

*Q-learning*, that initializes the estimates to large values and then follows a greedy policy. Optimistic Q-learning is widely used in applications and has been recognized as having good convergence in practice [10].

We prove that optimistic Q-learning, with the right setting of initial values, converge to a near optimal policy. This is not the first theoretical result showing that optimism helps in reinforcement learning, however previous results where concern with model based methods [7, 3]. We show the convergence of the widely used optimistic Q-learning, thus explaining and supporting the results observed in practice.

Our second result is a new and novel deterministic algorithm *incremental Q-learning*, which gradually promotes the values of actions that are not taken. We show that the frequency of sub-optimal actions vanishes, in the limit, and that the strategy defined by incremental Q-learning converges, in the limit, to the optimal policy (rather than a near optimal policy). Another view of incremental Q-learning is as a derandomization of the $\epsilon$-greedy Q-learning. The $\epsilon$-greedy Q-learning performs the sub optimal action every $1/\epsilon$ times in expectation, while the incremental Q-learning performs sub optimal action every $(Q(s, a(s)) - Q(s, b))/\epsilon$ times. Furthermore, by taking the appropriate values it can be similar to the Boltzman machine.

## 2 The Model

We define a Markov Decision process (MDP) as follows

**Definition 2.1** *A Markov Decision process (MDP) M is a 4-tuple $(S, A, P, R)$, where $S$ is a set of the states, $A$ is a set of actions, $P_{i,j}^M(a)$ is the transition probability from state $i$ to state $j$ when performing action $a \in A$ in state $i$, and $R_M(s, a)$ is the reward received when performing action $a$ in state $s$.*

A strategy for an MDP assigns, at each time $t$, for each state $s$ a probability for performing action $a \in A$, given a history $F_{t-1} = \{s_1, a_1, r_1, ..., s_{t-1}, a_{t-1}, r_{t-1}\}$ which includes the states, actions and rewards observed until time $t - 1$. While executing a strategy $\pi$ we perform at time $t$ action $a_t$ in state $s_t$ and observe a reward $r_t$ (distributed according to $R_M(s, a)$), and the next state $s_{t+1}$ distributed according to $P_{s_t, s_{t+1}}^M(a_t)$. We combine the sequence of rewards to a single value called *return*, and our goal is to maximize the return. In this work we focus on *discounted return*, which has a parameter $\gamma \in (0, 1)$, and the discounted return of policy $\pi$ is $V_M^\pi = \sum_{t=0}^\infty \gamma^t r_t$, where $r_t$ is the reward observed at time $t$.

We assume that $R_M(s, a)$ is non-negative and bounded by $R_{max}$, i.e, $\forall s, a : \ 0 \leq R_M(s, a) \leq R_{max}$. This implies that the discounted return is bounded by $V_{max} = \frac{R_{max}}{1-\gamma}$.

We define a value function for each state $s$, under policy $\pi$, as $V_M^\pi(s) = E[\sum_{i=0}^\infty r_i \gamma^i]$, where the expectation is over a run of policy $\pi$ starting at state $s$, and a state-action value function $Q_M^\pi(s, a) = E[R_M(s, a)] + \gamma \sum_{s'} P_{s,s'}^M(a) V_M^\pi(s')$.

Let $\pi^*$ be an optimal policy which maximizes the return from any start state. This implies that for any policy $\pi$ and any state $s$ we have $V_M^{\pi^*}(s) \geq V_M^\pi(s)$, and $\pi^*(s) = argmax_a(E[R_M(s, a)] + \gamma(\sum_{s'} P_{s,s'}^M(a) V^*(s'))$.

We use $V_M^*$ and $Q_M^*$ for $V_M^{\pi^*}$ and $Q_M^{\pi^*}$, respectively. We say that a policy $\pi$ is an $\epsilon$-optimal if $\|V_M^* - V_M^\pi\|_\infty \leq \epsilon$.

Given a trajectory let $T^{s,a}$ be the set of times in which we perform action $a$ in state $s$, $T^s = \cup_a T^{s,a}$ be the times when state $s$ is visited, $T^{s,not(a)} = T^s \setminus T^{s,a}$ be the set of times where in state $s$ an action $a' \neq a$ is performed, and $T^{not(s)} = \cup_{s' \neq s} T^{s'}$ be the set of times in which a state $s' \neq s$ is visited. Also, $[\#(s,a,t)]$ is the number of times action $a$ is performed in state $s$ up to time $t$, i.e., $|T^{s,a} \cap [1,t]|$.

Finally, throughout the paper we assume that the MDP is a uni-chain (see [9]), namely that from every state we can reach any other state.

## 3  Q-Learning

The Q-Learning algorithm [13] estimates the state-action value function (for discounted return) as follows:

$$Q_{t+1}(s,a) = (1 - \alpha_t(s,a))Q_t(s,a) + \alpha_t(s,a)(r_t(s,a) + \gamma V_t(s'))$$

where $s'$ is the state reached from state $s$ when performing action $a$ at time $t$, and $V_t(s) = \max_a Q_t(s,a)$. We assume that $\alpha_t(s',a') = 0$ for $t \notin T^{s',a'}$.

A learning rate $\alpha_t$ is *well-behaved* if for every state action pair $(s,a)$: (1) $\sum_{t=1}^{\infty} \alpha_t(s,a) = \infty$ and (2) $\sum_{t=1}^{\infty} \alpha_t^2(s,a) < \infty$. If the learning rate is well-behaved and every state action pair is performed infinitely often then Q-Learning converges to $Q^*$ with probability 1 (see [1, 11, 12, 8, 6]).

The convergence of Q-learning holds using any exploration policy, and only requires that each state action pair is executed infinitely often. The *greedy policy* with respect to the Q-values tries to exploit continuously, however, since it does not explore properly, it might result in poor return. At the other extreme *random policy* continuously explores, but its actual return may be very poor. An interesting compromise between the two extremes is the $\epsilon$-greedy policy, which is widely used in practice [10]. This policy executes the greedy policy with probability $1 - \epsilon$ and the random policy with probability $\epsilon$. This balance between exploration and exploitation both guarantees convergence and often good performance.

Common to many of the exploration techniques, is the use of randomization, which is also a very natural choice. In this work we explore strategies which perform exploration but avoids randomization and uses deterministic strategies.

## 4  Optimistic Q-Learning

*Optimistic Q-learning* is a simple greedy algorithm with respect to the Q-values, where the initial Q-values are set to large values, larger than their optimal values. We show that optimistic Q-learning converges to an $\epsilon$-optimal policy if the initial Q-values are set sufficiently large.

Let $\beta_\tau = \prod_{i=1}^{\tau}(1 - \alpha_i)$. We set the initial conditions of the Q-values as follows:

$$\forall s,a : \quad Q_0(s,a) = \frac{1}{\beta_T}V_{max},$$

where $T = T(\epsilon, \delta, S, A, \vec{\alpha})$ will be specified later. Let $\eta_{i,\tau} = \alpha_i \prod_{j=i+1}^{\tau}(1 - \alpha_j) = \alpha_i \beta_\tau / \beta_i$. Note that

$$Q_{t+1}(s,a) = (1 - \alpha_t)Q_t(s,a) + \alpha_t(r_t + \gamma V_t(s')) = \beta_\tau Q_0(s,a) + \sum_{i=1}^{\tau} \eta_{i,\tau} r_i(s,a) + \gamma \sum_{i=1}^{\tau} \eta_{i,\tau} V_{t_i}(s_i),$$

where $\tau = [\#(s,a,t)]$ and $s_i$ is the next state arrived at time $t_i$ when action $a$ is performed for the $i$th time in state $s$.

First we show that as long as $\tau = [\#(s,a,t)] \leq T$ actions $a$ are performed in state $s$, we have $Q_t(s,a) \geq V_{max}$. Latter we will use this to show that action $a$ is performed at least $T$ times in state $s$.

**Lemma 4.1** *In optimistic Q-learning for any state $s$, action $a$ and time $t$, such that $\tau = [\#(s,a,t)] \leq T$ we have $Q_t(s,a) \geq V_{max} \geq Q^*(s,a)$.*

Lemma 4.1 follows from the following observation:

$$Q_t(s,a) = \beta_\tau Q_0(s,a) + \sum_{i=1}^{\tau} \eta_{i,\tau} r_i(s,a) + \gamma \sum_{i=1}^{\tau} \eta_{i,\tau} V_{t_i}(s_i) \geq \frac{\beta_\tau}{\beta_T} V_{max} \geq V^*(s).$$

Now we bound $T$ as a function of the algorithm parameters (i.e., $\epsilon, \delta, |S|, |A|$) and the learning rate. We need to set $T$ large enough to guarantee that with probability $1 - \delta$, for any $t > T$ updates, using the given learning rate, the deviation from the true value is at most $\epsilon$. Formally, given a sequence $X_t$ of i.i.d. random variables with zero mean and bounded by $V_{max}$, and a learning rate $\alpha_t = (1/[\#(s,a,t)])^\omega$ let $Z_{t+1} = (1 - \alpha_t)Z_t + \alpha_t X_t$. A time $T(\epsilon, \delta)$ is an *initialization time* if $Pr[\forall t \geq T : Z_t \leq \epsilon] \geq 1 - \delta$. The following lemma bounds the initialization time as a function of the parameter $\omega$ of the learning rate.

**Lemma 4.2** *The initialization time for $\vec{X}$ and $\vec{\alpha}$ is at most $T(\epsilon, \delta) = c\left(\left(\frac{V_{max}^2}{\epsilon^2}(ln(1/\delta) + ln(V_{max}/\epsilon))\right)^{\frac{1}{\omega}}\right)$, for some constant $c$.*

We define a modified process, in which we update using the optimal value function, rather than our current estimate. For $t \geq 1$ we have,

$$\hat{Q}_{t+1}(s,a) = (1 - \alpha_t(s,a))\hat{Q}_t(s,a) + \alpha_t(s,a)(r_t(s,a) + \gamma V^*(s')),$$

where $s'$ is the next state. The following lemma bounds the difference between $Q^*$ and $\hat{Q}_t$.

**Lemma 4.3** *Consider optimistic Q-learning and let $T = T(\epsilon, \delta)$ be the initialization time. Then with probability $1 - \delta$, for any $t > T$, we have $Q^*(s,a) - \hat{Q}(s,a) \leq \epsilon$.*

**Proof:** Let $\tau = [\#(s,a,t)]$. By definition we have

$$\hat{Q}_t(s,a) = \beta_\tau Q_0(s,a) + \sum_{i=1}^{\tau} \eta_{i,\tau} r_i + \gamma \sum_{i=1}^{\tau} \eta_{i,\tau} V^*(s_i).$$

This implies that,

$$Q^*(s,a) - \hat{Q}(s,a) = -\beta_\tau Q_0(s,a) + error\_r[s,a,t] + error\_v[s,a,t]$$

where $error\_r[s,a,t] = E[R(s,a)] - \sum_{i=1}^{\tau} \eta_{i,\tau} r_i$, and $error\_v[s,a,t] = E[V^*(s')|s,a] - \sum_{i=1}^{\tau} \eta_{i,\tau} V^*(s_i)$. We bound both $error\_r[s,a,t]$ and $error\_v[s,a,t]$ using Lemma 4.2. Therefore, with probability $1 - \delta$, we have $Q^*(s,a) - \hat{Q}(s,a) \leq \epsilon$, for any $t \geq T$. **Q.E.D.**

Next we bound the difference between our estimate $V_t(s)$ and $V^*(s)$.

**Lemma 4.4** *Consider optimistic Q-learning and let $T = T((1-\gamma)\epsilon, \delta/|S||A|)$ be the initialization time. With probability at least $1 - \delta$ for any state $s$ and time $t$, we have $V^*(s) - V_t(s) \leq \epsilon$.*

**Proof:** By Lemma 4.3 we have that with probability $1 - \delta$ for every state $s$, action $a$ and time $t$ we have $Q^*(s,a) - \hat{Q}_t(s,a) \leq (1-\gamma)\epsilon$. We show by induction on $t$ that $V^*(s) - V_t(s) \leq \epsilon$, for every state $s$. For $t = 0$ we have $V_0(s) > V_{max}$ and hence the claim holds. For the inductive step assume it holds up to time $t$ and show that it hold for time $t + 1$. Let $(s, a)$ be the state action pair executed in time $t + 1$. If $[\#(s,a,t+1)] \leq T$ then by Lemma 4.1, $V_t(s) \geq V_{max} \geq V^*(s)$, and the induction claim holds. Otherwise, let $a^*$ be the optimal action at state $s$, then,

$$
\begin{aligned}
V^*(s) - V_{t+1}(s) &\leq Q^*(s, a^*) - Q_{t+1}(s, a^*) \\
&= Q^*(s, a^*) - \hat{Q}_{t+1}(s, a^*) + \hat{Q}_{t+1}(s, a^*) - Q_{t+1}(s, a^*) \\
&\leq (1-\gamma)\epsilon + \gamma \sum_{i=1}^{\tau} \eta_{i,\tau}(V^*(s_i) - V_{t_i}(s_i)),
\end{aligned}
$$

where $\tau = [\#(s, a, t)]$, $t_i$ is the time when the $i$-th time the action $a$ is performed in state $s$, and state $s_i$ is the next state. Since $t_i \leq t$, by the inductive hypothesis we have that $V^*(s_i) - V_{t_i}(s_i) \leq \epsilon$, and therefore,

$$
V^*(s) - V_{t+1}(s) \leq (1-\gamma)\epsilon + \gamma\epsilon = \epsilon.
$$

**Q.E.D.**

**Lemma 4.5** *Consider optimistic Q-learning and let $T = T((1-\gamma)\epsilon, \delta/|S||A|)$ be the initialization time. With probability at least $1 - \delta$ any state action pair $(s, a)$ that is executed infinitely often is $\epsilon$-optimal, i.e., $V^*(s) - Q^*(s, a) \leq \epsilon$.*

**Proof:** Given a trajectory let $U'$ be the set of state action pairs that are executed infinitely often, and let $M'$ be the original MDP $M$ restricted to $U'$. For $M'$ we can use the classical convergence proofs, and claim that $V_t(s)$ converges to $V^*_{M'}(s)$ and $Q_t(s, a)$, for $(s, a) \in U'$, converges to $Q^*_{M'}(s, a)$, both with probability 1. Since $(s, a) \in U'$ is performed infinitely often it implies that $Q_t(s, a)$ converges to $V_t(s) = V^*_{M'}(s)$ and therefore $Q^*_{M'}(s, a) = V^*_{M'}(s)$. By Lemma 4.4 with probability $1 - \delta$ we have that $V^*_M(s) - V_t(s) \leq \epsilon$, therefore $V^*_M(s) - Q^*_M(s, a) \leq V^*_M(s) - Q^*_{M'}(s, a) \leq \epsilon$. **Q.E.D.**

A simple corollary is that if we set $\epsilon$ small enough, e.g., $\epsilon < \min_{(s,a)}\{V^*(s) - Q^*(s,a)|V^*(s) \neq Q^*(s,a)\}$, then optimistic Q-learning converges to the optimal policy. Another simple corollary is the following theorem.

**Theorem 4.6** *Consider optimistic Q-learning and let $T = T((1-\gamma)\epsilon, \delta/|S||A|)$ be the initialization time. For any constant $\xi$, with probability at least $1 - \delta$ there is a time $T_\xi > T$ such that at any time $t > T_\xi$ the strategy defined by the optimistic Q-learning is $(\epsilon + \xi)/(1-\gamma)$-optimal.*

## 5 Incremental Q-learning

In this section we describe a new algorithm that we call *incremental Q-learning*. The main idea of the algorithm is to achieve a deterministic tradeoff between exploration and exploitation.

Incremental Q-learning is a greedy policy with respect to the estimated Q-values plus a promotion term. The promotion term of a state-action pair $(s, a)$ is promoted

each time the action $a$ is *not* executed in state $s$, and zeroed each time action $a$ is executed. We show that in incremental Q-learning every state-action pair is taken infinitely often, which implies standard convergence of the estimates. We show that the fraction of time in which sub-optimal actions are executed vanishes in the limit. This implies that the strategy defined by incremental Q-learning converges, in the limit, to the optimal policy. Incremental Q-learning estimates the Q-function as in Q-learning:

$$Q_{t+1}(s,a) = (1 - \alpha_t(s,a))Q_t(s,a) + \alpha_t(s,a)(r_t(s,a) + \gamma V_t(s'))$$

where $s'$ is the next state reached when performing action $a$ in state $s$ at time $t$. The promotion term $A_t$ is define as follows:

$$A_{t+1}(s,a) = 0 : \quad t \in T^{s,a}$$
$$A_{t+1}(s,a) = A_t(s,a) + \psi([\#(s,a,t)]) : \quad t \in T^{s,not(a)}$$
$$A_{t+1}(s,a) = A_t(s,a) : \quad t \in T^{not(s)},$$

where $\psi(i)$ is a *promotion function* which in our case depends only on the number of times we performed $(s,a')$, $a' \neq a$, since the last time we performed $(s,a)$. We say that a promotion function $\psi$ is well-behaved if: (1) The function $\psi$ converges to zero, i.e., $\lim_{i \to \infty} \psi(i) = 0$, and (2) $\psi(1) = 1$ and $\psi(k) > \psi(k+1) > 0$. For example $\psi(i) = \frac{1}{i}$ is well behaved promotion function.

Incremental Q-learning is a greedy policy with respect to $S_t(s,a) = Q_t(s,a) + A_t(s,a)$. First we show that $Q_t$, in incremental Q-learning, converges to $Q^*$.

**Lemma 5.1** *Consider incremental Q-learning using a well-behaved learning rate and a well-behaved promotion function. Then $Q_t$ converges to $Q^*$ with probability 1.*

**Proof:** Since the learning rate is well-behaved, we need only to show that each state action pair is performed infinitely often. We show that each state that is visited infinitely often, all of its actions are performed infinitely often. Since the MDP is uni-chain this will imply that with probability 1 we reach all states infinitely often, which completes the proof.

Assume that state $s$ is visited infinitely often. Since $s$ is visited infinitely often, there has to be a non-empty subset of the actions $A'$ which are performed infinitely often in $s$. The proof is by contradiction, namely assume that $A' \neq A$. Let $t_1$ be the last time that an action not in $A'$ is performed in state $s$. Since $\psi$ is well behaved we have that $\psi(t_1)$ is constant for a fixed $t_1$, it implies that $A_t(s,a)$ diverges for $a \notin A'$. Therefore, eventually we reach a time $t_2 > t_1$ such that $A_{t_2}(s,a) > V_{max}$, for every $a \notin A'$. Since the actions in $A'$ are performed infinitely often there is a time $t_3 > t_2$ such that each action $a' \in A'$ is performed at least once in $[t_2, t_3]$. This implies that $A_{t_3}(s,a) > V_{max} + A_{t_3}(s,a')$ for any $a' \in A'$ and $a \notin A'$. Therefore, some action in $a \in A \setminus A'$ will be performed after $t_1$, contradicting our assumption. **Q.E.D.**

The following lemma shows that the frequency of sub-optimal actions vanishes.

**Lemma 5.2** *Consider incremental Q-learning using a well behaved learning rate and a well behaved promotion function. Let $f_t(s,a) = |T^{s,a}|/|T^s|$ and $(s,a)$ be any sub-optimal state-action pair. Then $\lim_{t \to \infty} f_t(s,a) = 0$, with probability 1.*

The intuition behind Lemma 5.2 is the following. Let $a^*$ be an optimal action in state $s$ and $a$ be a sub-optimal action. By Lemma 5.1, with probability 1 both

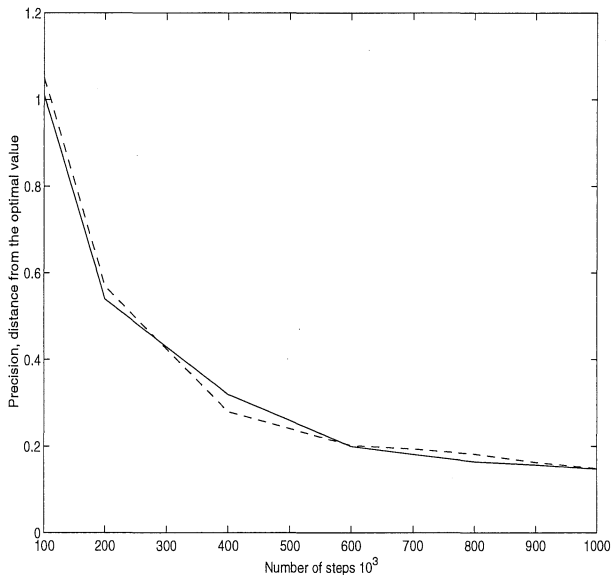

Figure 1: Example of 50 states MDP, where the discount factor, $\gamma$, is 0.9. The leaning rate of both Incremantal and epsilon greedy Q-learning is set to 0.8. The dashed line represents the epsilon greedy Q-learning.

$Q_t(s, a^*)$ converges to $Q^*(s, a^*) = V^*(s)$ and $Q_t(s, a)$ converges to $Q^*(s, a)$. This implies, intuitively, that $A_t(s, a)$ has to be at least $V^*(s) - Q^*(s, a) = h > 0$ for $(s, a)$ to be executed. Since the promotion function is well behaved, the number of time steps required until $A_t(s, a)$ changes from 0 to $h$ increases after each time we perform $(s, a)$. Since the inter-time between executions of $(s, a)$ diverges, the frequency $f_t(s, a)$ vanishes.

The following corollary gives a quantitative bound.

**Corollary 5.3** *Consider incremental Q-learning with learning rate* $\alpha_t(s, a) = 1/[\#(s, a, t)]$ *and* $\psi(k) = 1/e^k$. *Let* $(s, a)$ *be a sub-optimal state-action pair. The number of times* $(s, a)$ *is performed in the first* $n$ *visits to state* $s$ *is* $\Theta(\frac{\ln(n)}{V^*(s) - Q^*(s, a)})$, *for sufficiently large* $n$.

Furthermore, the return obtained by incremental Q-learning converges to the optimal return.

**Corollary 5.4** *Consider incremental Q-learning using a well behaved learning rate and a well behaved promotion function. For every* $\epsilon$ *there exists a time* $T_\epsilon$ *such that for any* $t > T_\epsilon$ *we have that the strategy* $\pi$ *defined by incremental Q-learning is* $\epsilon$-*optimal with probability* 1.

# 6 Experiments

In this section we show some experimental results, comparing Incremental Q-Learning and epsilon-greedy Q-Learning. One can consider incremental Q-learning as a derandomization of $\epsilon_t$-greedy Q-Learning, where the promotion function satisfies $\psi_t = \epsilon_t$.

The experiment was made on MDP, which includes 50 states and two actions per state. Each state action pair immediate reward is randomly chosen in the interval $[0, 10]$. For each state and action $(s, a)$ the next state transition is random, i.e., for every state $s'$ we have a random variable $X_{s'}^{s,a} \in [0, 1]$ and $P_{s,s'}^a = \frac{X_{s'}^{s,a}}{\sum_{\tilde{s}} X_{\tilde{s}}^{s,a}}$. For the $\epsilon_t$-greedy Q-learning, we have $\epsilon_t = 10000/t$ at time $t$, while for the incremental we have $\psi_t = 10000/t$. Each result in the experiment is an average of ten different runs. In Figure 1, we observe similar behavior of the two algorithms. This experiment demonstrates the strong experimental connection between these methods. We plan to further investigate the theoretical connection between $\epsilon$-greedy, Boltzman machine and incremental Q-Learning.

## 7    Acknowledgements

This research was supported in part by a grant from the Israel Science Foundation.

## References

[1] D. P. Bertsekas and J. N. Tsitsiklis. *Neuro-Dynamic Programming*. Athena Scientific, Belmont, MA, 1996.

[2] V.S. Borkar and S.P. Meyn. The O.D.E. method for convergence of stochastic approximation and reinforcement learning. *Siam J. control*, 38 (2):447–69, 2000.

[3] R. I. Brafman and M. Tennenholtz. R-max - a general polynomial time algorithm for near-optimal reinforcement learning. In *IJCAI*, 2001.

[4] E. Even-Dar and Y. Mansour. Learning rates for Q-learning. In *COLT*, 2001.

[5] J. C. Gittins and D. M. Jones. A dynamic allocation index for the sequential design of experiments. *Progress in Statistics*, pages 241 –266, 1974.

[6] T. Jaakkola, M. I. Jordan, and S. P. Singh. On the convergence of stochastic iterative dynamic programming algorithms. *Neural Computation, 6*, 1994.

[7] M. Kearns and S. Singh. Efficient reinforcement learning: theoretical framework and algorithms. In *ICML*, 1998.

[8] M. Littman and Cs. Szepesvari. A generalized reinforcement learning model: convergence and applications. In *ICML*, 1996.

[9] M.L Puterman. *Markov Decision Processes - Discrete Stochastic Dynamic Programming*. John Wiley & Sons. Inc., New York, NY, 1994.

[10] R. S. Sutton and A. G. Bato. *Reinforcement Learning*. MIT press, 1998.

[11] J. N. Tsitsiklis. Asynchronous stochastic approximation and Q-learning. *Machine Learning*, 16:185–202, 1994.

[12] C. Watkins and P. Dyan. Q-learning. *Machine Learning*, 8(3/4):279 –292, 1992.

[13] C. Watkins. *Learning from Delayed Rewards*. PhD thesis, Cambridge University, 1989.
